# Sparse Feature Learning for Deep Belief Networks

**Marc'Aurelio Ranzato**[1]        **Y-Lan Boureau**[2,1]        **Yann LeCun**[1]
[1] Courant Institute of Mathematical Sciences, New York University
[2] INRIA Rocquencourt
{ranzato,ylan,yann@courant.nyu.edu}

## Abstract

Unsupervised learning algorithms aim to discover the structure hidden in the data, and to learn representations that are more suitable as input to a supervised machine than the raw input. Many unsupervised methods are based on reconstructing the input from the representation, while constraining the representation to have certain desirable properties (e.g. low dimension, sparsity, etc). Others are based on approximating density by stochastically reconstructing the input from the representation. We describe a novel and efficient algorithm to learn sparse representations, and compare it theoretically and experimentally with a similar machine trained probabilistically, namely a Restricted Boltzmann Machine. We propose a simple criterion to compare and select different unsupervised machines based on the trade-off between the reconstruction error and the information content of the representation. We demonstrate this method by extracting features from a dataset of handwritten numerals, and from a dataset of natural image patches. We show that by stacking multiple levels of such machines and by training sequentially, high-order dependencies between the input observed variables can be captured.

## 1   Introduction

One of the main purposes of unsupervised learning is to produce *good representations* for data, that can be used for detection, recognition, prediction, or visualization. Good representations eliminate irrelevant variabilities of the input data, while preserving the information that is useful for the ultimate task. One cause for the recent resurgence of interest in unsupervised learning is the ability to produce *deep feature hierarchies* by stacking unsupervised modules on top of each other, as proposed by Hinton et al. [1], Bengio et al. [2] and our group [3, 4]. The unsupervised module at one level in the hierarchy is fed with the representation vectors produced by the level below. Higher-level representations capture high-level dependencies between input variables, thereby improving the ability of the system to capture underlying regularities in the data. The output of the last layer in the hierarchy can be fed to a conventional supervised classifier.

A natural way to design stackable unsupervised learning systems is the *encoder-decoder* paradigm [5]. An *encoder* transforms the input into the representation (also known as the *code* or the feature vector), and a *decoder* reconstructs the input (perhaps stochastically) from the representation. PCA, Auto-encoder neural nets, Restricted Boltzmann Machines (RBMs), our previous sparse energy-based model [3], and the model proposed in [6] for noisy overcomplete channels are just examples of this kind of architecture. The encoder/decoder architecture is attractive for two reasons: 1. after training, computing the code is a very fast process that merely consists in running the input through the encoder; 2. reconstructing the input with the decoder provides a way to check that the code has captured the relevant information in the data. Some learning algorithms [7] do not have a decoder and must resort to computationally expensive Markov Chain Monte Carlo (MCMC) sampling methods in order to provide reconstructions. Other learning algorithms [8, 9] lack an encoder, which makes it necessary to run an expensive optimization algorithm to find the code associated with each new input sample. In this paper we will focus only on encoder-decoder architectures.

In general terms, we can view an unsupervised model as defining a distribution over input vectors $Y$ through an energy function $E(Y, Z, W)$:

$$P(Y|W) = \int_z P(Y, z|W) = \frac{\int_z e^{-\beta E(Y,z,W)}}{\int_{y,z} e^{-\beta E(y,z,W)}} \tag{1}$$

where $Z$ is the code vector, $W$ the trainable parameters of encoder and decoder, and $\beta$ is an arbitrary positive constant. The energy function includes the *reconstruction error*, and perhaps other terms as well. For convenience, we will omit $W$ from the notation in the following. Training the machine to model the input distribution is performed by finding the encoder and decoder parameters that minimize a loss function equal to the negative log likelihood of the training data under the model. For a single training sample $Y$, the loss function is

$$L(W, Y) = -\frac{1}{\beta} \log \int_z e^{-\beta E(Y,z)} + \frac{1}{\beta} \log \int_{y,z} e^{-\beta E(y,z)} \tag{2}$$

The first term is the *free energy* $F_\beta(Y)$. Assuming that the distribution over $Z$ is rather peaked, it can be simpler to approximate this distribution over $Z$ by its mode, which turns the marginalization over $Z$ into a minimization:

$$L^*(W, Y) = E(Y, Z^*(Y)) + \frac{1}{\beta} \log \int_y e^{-\beta E(y, Z^*(y))} \tag{3}$$

where $Z^*(Y)$ is the maximum likelihood value $Z^*(Y) = \operatorname{argmin}_z E(Y, z)$, also known as the *optimal code*. We can then define an energy for each input point, that measures how well it is reconstructed by the model:

$$F_\infty(Y) = E(Y, Z^*(Y)) = \lim_{\beta \to \infty} -\frac{1}{\beta} \log \int_z e^{-\beta E(Y,z)} \tag{4}$$

The second term in equation 2 and 3 is called the *log partition function*, and can be viewed as a penalty term for low energies. It ensures that the system produces low energy *only* for input vectors that have high probability in the (true) data distribution, and produces higher energies for all other input vectors [5]. The overall loss is the average of the above over the training set.

Regardless of whether only $Z^*$ or the whole distribution over $Z$ is considered, the main difficulty with this framework is that it can be very hard to compute the gradient of the log partition function in equation 2 or 3 with respect to the parameters $W$. Efficient methods shortcut the computation by drastically and cleverly reducing the integration domain. For instance, Restricted Boltzmann Machines (RBM) [10] approximate the gradient of the log partition function in equation 2 by *sampling* values of $Y$ whose energy will be pulled up using an MCMC technique. By running the MCMC for a short time, those samples are chosen in the vicinity of the training samples, thereby ensuring that the energy surface forms a ravine around the manifold of the training samples. This is the basis of the Contrastive Divergence method [10].

The role of the log partition function is merely to ensure that the energy surface is lower around training samples than anywhere else. The method proposed here eliminates the log partition function from the loss, and replaces it by a term that *limits the volume of the input space over which the energy surface can take a low value*. This is performed by *adding a penalty term on the code rather than on the input*. While this class of methods does not directly maximize the likelihood of the data, it can be seen as a crude approximation of it. To understand the method, we first note that if for each vector $Y$, there exists a corresponding optimal code $Z^*(Y)$ that makes the reconstruction error (or energy) $F_\infty(Y)$ zero (or near zero), the model can perfectly reconstruct any input vector. This makes the energy surface flat and indiscriminate. On the other hand, if $Z$ can only take a small number of different values (low entropy code), then the energy $F_\infty(Y)$ can only be low in a limited number of places (the $Y$'s that are reconstructed from this small number of $Z$ values), and the energy cannot be flat.

More generally, a convenient method through which flat energy surfaces can be avoided is to *limit the maximum information content of the code*. Hence, *minimizing the energy $F_\infty(Y)$ together with the information content of the code is a good substitute for minimizing the log partition function*.

A popular way to minimize the information content in the code is to make the code sparse or low-dimensional [5]. This technique is used in a number of unsupervised learning methods, including PCA, auto-encoders neural network, and sparse coding methods [6, 3, 8, 9]. In sparse methods, the code is forced to have only a few non-zero units while most code units are zero most of the time. Sparse-overcomplete representations have a number of theoretical and practical advantages, as demonstrated in a number of recent studies [6, 8, 3]. In particular, they have good robustness to noise, and provide a good tiling of the joint space of location and frequency. In addition, they are advantageous for classifiers because classification is more likely to be easier in higher dimensional spaces. This may explain why biology seems to like sparse representations [11]. In our context, the main advantage of sparsity constraints is to allow us to replace a marginalization by a minimization, and to free ourselves from the need to minimize the log partition function explicitly.

In this paper we propose a new unsupervised learning algorithm called Sparse Encoding Symmetric Machine (SESM), which is based on the encoder-decoder paradigm, and which is able to produce sparse overcomplete representations efficiently without any need for filter normalization [8, 12] or code saturation [3]. As described in more details in sec. 2 and 3, we consider a loss function which is a weighted sum of the reconstruction error and a sparsity penalty, as in many other unsupervised learning algorithms [13, 14, 8]. Encoder and decoder are constrained to be *symmetric*, and share a set of linear filters. Although we only consider linear filters in this paper, the method allows the use of any differentiable function for encoder and decoder. We propose an iterative on-line learning algorithm which is closely related to those proposed by Olshausen and Field [8] and by us previously [3]. The first step computes the optimal code by minimizing the energy for the given input. The second step updates the parameters of the machine so as to minimize the energy.

In sec. 4, we compare SESM with RBM and PCA. Following [15], we evaluate these methods by measuring the reconstruction error for a given entropy of the code. In another set of experiments, we train a classifier on the features extracted by the various methods, and measure the classification error on the MNIST dataset of handwritten numerals. Interestingly, the machine achieving the best recognition performance is the one with the best trade-off between RMSE and entropy. In sec. 5, we compare the filters learned by SESM and RBM for handwritten numerals and natural image patches. In sec.5.1.1, we describe a simple way to produce a deep belief net by stacking multiple levels of SESM modules. The representational power of this hierarchical non-linear feature extraction is demonstrated through the *unsupervised* discovery of the numeral class labels in the high-level code.

## 2   Architecture

In this section we describe a Sparse Encoding Symmetric Machine (SESM) having a set of linear filters in both encoder and decoder. However, everything can be easily extended to any other choice of parameterized functions as long as these are differentiable and maintain symmetry between encoder and decoder. Let us denote with $Y$ the input defined in $R^N$, and with $Z$ the code defined in $R^M$, where $M$ is in general greater than $N$ (for overcomplete representations). Let the filters in encoder and decoder be the columns of matrix $W \in R^{N \times M}$, and let the biases in the encoder and decoder be denoted by $b_{enc} \in R^M$ and $b_{dec} \in R^N$, respectively. Then, encoder and decoder compute:

$$f_{enc}(Y) = W^T Y + b_{enc}, \qquad f_{dec}(Z) = Wl(Z) + b_{dec} \qquad (5)$$

where the function $l$ is a point-wise logistic non-linearity of the form:

$$l(x) = 1/(1 + exp(-gx)), \qquad (6)$$

with $g$ fixed gain. The system is characterized by an energy measuring the compatibility between pairs of input $Y$ and latent code $Z$, $E(Y, Z)$ [16]. The lower the energy, the more compatible (or likely) is the pair. We define the energy as:

$$E(Y, Z) = \alpha_e \|Z - f_{enc}(Y)\|_2^2 + \|Y - f_{dec}(Z)\|_2^2 \qquad (7)$$

During training we minimize the following loss:

$$
\begin{aligned}
L(W, Y) &= E(Y, Z) + \alpha_s h(Z) + \alpha_r \|W\|_1 \\
&= \alpha_e \|Z - f_{enc}(Y)\|_2^2 + \|Y - f_{dec}(Z)\|_2^2 + \alpha_s h(Z) + \alpha_r \|W\|_1 \qquad (8)
\end{aligned}
$$

The first term tries to make the output of the encoder as similar as possible to the code $Z$. The second term is the mean-squared error between the input $Y$ and the reconstruction provided by the decoder.

The third term ensures the *sparsity* of the code by penalizing non zero values of code units; this term acts independently on each code unit and it is defined as $h(Z) = \sum_{i=1}^{M} \log(1 + l^2(z_i))$, (corresponding to a factorized Student-t prior distribution on the non linearly transformed code units [8] through the logistic of equation 6). The last term is an L1 regularization on the filters to suppress noise and favor more localized filters. The loss formulated in equation 8 combines terms that characterize also other methods. For instance, the first two terms appear in our previous model [3], but in that work, the weights of encoder and decoder were not tied and the parameters in the logistic were updated using running averages. The second and third terms are present in the "decoder-only" model proposed in [8]. The third term was used in the "encoder-only" model of [7]. Besides the already-mentioned advantages of using an encoder-decoder architecture, we point out another good feature of this algorithm due to its symmetry. A common idiosyncrasy for sparse-overcomplete methods using both a reconstruction and a sparsity penalty in the objective function (second and third term in equation 8), is the need to *normalize* the basis functions in the decoder during learning [8, 12] with somewhat ad-hoc technique, otherwise some of the basis functions collapse zero, and some blow up to infinity. Because of the sparsity penalty and the linear reconstruction, code units become tiny and are compensated by the filters in the decoder that grow without bound. Even though the overall loss decreases, training is unsuccessful. Unfortunately, simply normalizing the filters makes less clear which objective function is minimized. Some authors have proposed quite expensive methods to solve this issue: by making better approximations of the posterior distribution [15], or by using sampling techniques [17]. In this work, we propose to enforce *symmetry* between encoder and decoder (through weight sharing) so as to have automatic scaling of filters. Their norm cannot possibly be large because code units, produced by the encoder weights, would have large values as well, producing bad reconstructions and increasing the energy (the second term in equation 7 and 8).

## 3   Learning Algorithm

Learning consists of determining the parameters in $W$, $b_{enc}$, and $b_{dec}$ that minimize the loss in equation 8. As indicated in the introduction, the energy augmented with the sparsity constraint is minimized with respect to the code to find the optimal code. No marginalization over code distribution is performed. This is akin to using the loss function in equation 3. However, the log partition function term is dropped. Instead, we rely on the code sparsity constraints to ensure that the energy surface is not flat.

Since the second term in equation 8 couples both $Z$ and $W$ and $b_{dec}$, it is not straightforward to minimize this energy with respect to both. On the other hand, once $Z$ is given, the minimization with respect to $W$ is a convex quadratic problem. Vice versa, if the parameters $W$ are fixed, the optimal code $Z^*$ that minimizes $L$ can be computed easily through gradient descent. This suggests the following iterative on-line coordinate descent learning algorithm:
**1.** for a given sample $Y$ and parameter setting, minimize the loss in equation 8 with respect to $Z$ by gradient descent to obtain the optimal code $Z^*$
**2.** clamping both the input $Y$ and the optimal code $Z^*$ found at the previous step, do *one step* of gradient descent to update the parameters.
Unlike other methods [8, 12], no column normalization of $W$ is required. Also, all the parameters are updated by gradient descent unlike in our previous work [3] where some parameters are updated using a moving average.

After training, the system converges to a state where the decoder produces good reconstructions from a sparse code, and the optimal code is predicted by a simple feed-forward propagation through the encoder.

## 4   Comparative Coding Analysis

In the following sections, we mainly compare SESM with RBM in order to better understand their differences in terms of maximum likelihood approximation, and in terms of coding efficiency and robustness.

**RBM**     As explained in the introduction, RBMs minimize an approximation of the negative log likelihood of the data under the model. An RBM is a binary stochastic symmetric machine defined

by an energy function of the form: $E(Y, Z) = -Z^T W^T Y - b_{enc}^T Z - b_{dec}^T Y$. Although this is not obvious at first glance, this energy can be seen as a special case of the encoder-decoder architecture that pertains to binary data vectors and code vectors [5]. Training an RBM minimizes an approximation of the negative log likelihood loss function 2, averaged over the training set, through a gradient descent procedure. Instead of estimating the gradient of the log partition function, RBM training uses contrastive divergence [10], which takes random samples drawn over a limited region $\Omega$ around the training samples. The loss becomes:

$$L(W, Y) = -\frac{1}{\beta} \log \sum_z e^{-\beta E(Y, z)} + \frac{1}{\beta} \log \sum_{y \in \Omega} \sum_z e^{-\beta E(y, z)} \qquad (9)$$

Because of the RBM architecture, given a $Y$, the components of $Z$ are independent, hence the sum over configurations of $Z$ can be done independently for each component of $Z$. Sampling $y$ in the neighborhood $\Omega$ is performed with one, or a few alternated MCMC steps over $Y$, and $Z$. This means that only the energy of points around training samples is pulled up. Hence, the likelihood function takes the right shape around the training samples, but not necessarily everywhere. However, the code vector in an RBM is binary and noisy, and one may wonder whether this does not have the effect of surreptitiously limiting the information content of the code, thereby further minimizing the log partition function as a bonus.

**SESM**     RBM and SESM have almost the same architecture because they both have a symmetric encoder and decoder, and a logistic non-linearity on the top of the encoder. However, RBM is trained using (approximate) maximum likelihood, while SESM is trained by simply minimizing the average energy $F_\infty(Y)$ of equation 4 with an additional code sparsity term. SESM relies on the sparsity term to prevent flat energy surfaces, while RBM relies on an explicit contrastive term in the loss, an approximation of the log partition function. Also, the coding strategy is very different because code units are "noisy" and binary in RBM, while they are quasi-binary and *sparse* in SESM. Features extracted by SESM look like object parts (see next section), while features produced by RBM lack an intuitive interpretation because they aim at modeling the input distribution and they are used in a *distributed* representation.

## 4.1   Experimental Comparison

In the first experiment we have trained SESM, RBM, and PCA on the first 20000 digits in the MNIST training dataset [18] in order to produce codes with 200 components. Similarly to [15] we have collected test image codes after the logistic non linearity (except for PCA which is linear), and we have measured the root mean square error (RMSE) and the entropy. SESM was run for different values of the sparsity coefficient $\alpha_s$ in equation 8 (while all other parameters are left unchanged, see next section for details). The RMSE is defined as $\frac{1}{\sigma} \sqrt{\frac{1}{PN} \|Y - f_{dec}(\bar{Z})\|_2^2}$, where $\bar{Z}$ is the *uniformly quantized code* produced by the encoder, $P$ is the number of test samples, and $\sigma$ is the estimated variance of units in the input $Y$. Assuming to encode the (quantized) code units independently and with the same distribution, the lower bound on the number of bits required to encode each of them is given by: $H_{c.u.} = -\sum_{i=1}^{Q} \frac{c_i}{PM} \log_2 \frac{c_i}{PM}$, where $c_i$ is the number of counts in the $i$-th bin, and $Q$ is the number of quantization levels. The number of bits *per pixel* is then equal to: $\frac{M}{N} H_{c.u.}$. Unlike in [15, 12], the reconstruction is done taking the quantized code in order to measure the robustness of the code to the quantization noise. As shown in fig. 1-C, RBM is very robust to noise in the code because it is trained by sampling. The opposite is true for PCA which achieves the lowest RMSE when using high precision codes, but the highest RMSE when using a coarse quantization. SESM seems to give the best trade-off between RMSE and entropy. Fig. 1-D/F compare the features learned by SESM and RBM. Despite the similarities in the architecture, filters look quite different in general, revealing two different coding strategies: distributed for RBM, and sparse for SESM.

In the second experiment, we have compared these methods by means of a *supervised* task in order to assess which method produces the most discriminative representation. Since we have available also the labels in the MNIST, we have used the codes (produced by these machines trained unsupervised) as input to the *same* linear classifier. This is run for 100 epochs to minimize the squared error between outputs and targets, and has a mild ridge regularizer. Fig. 1-A/B show the result of these experiments in addition to what can be achieved by a linear classifier trained on the raw pixel data. Note that: 1) training on features instead of raw data improves the recognition (except for PCA

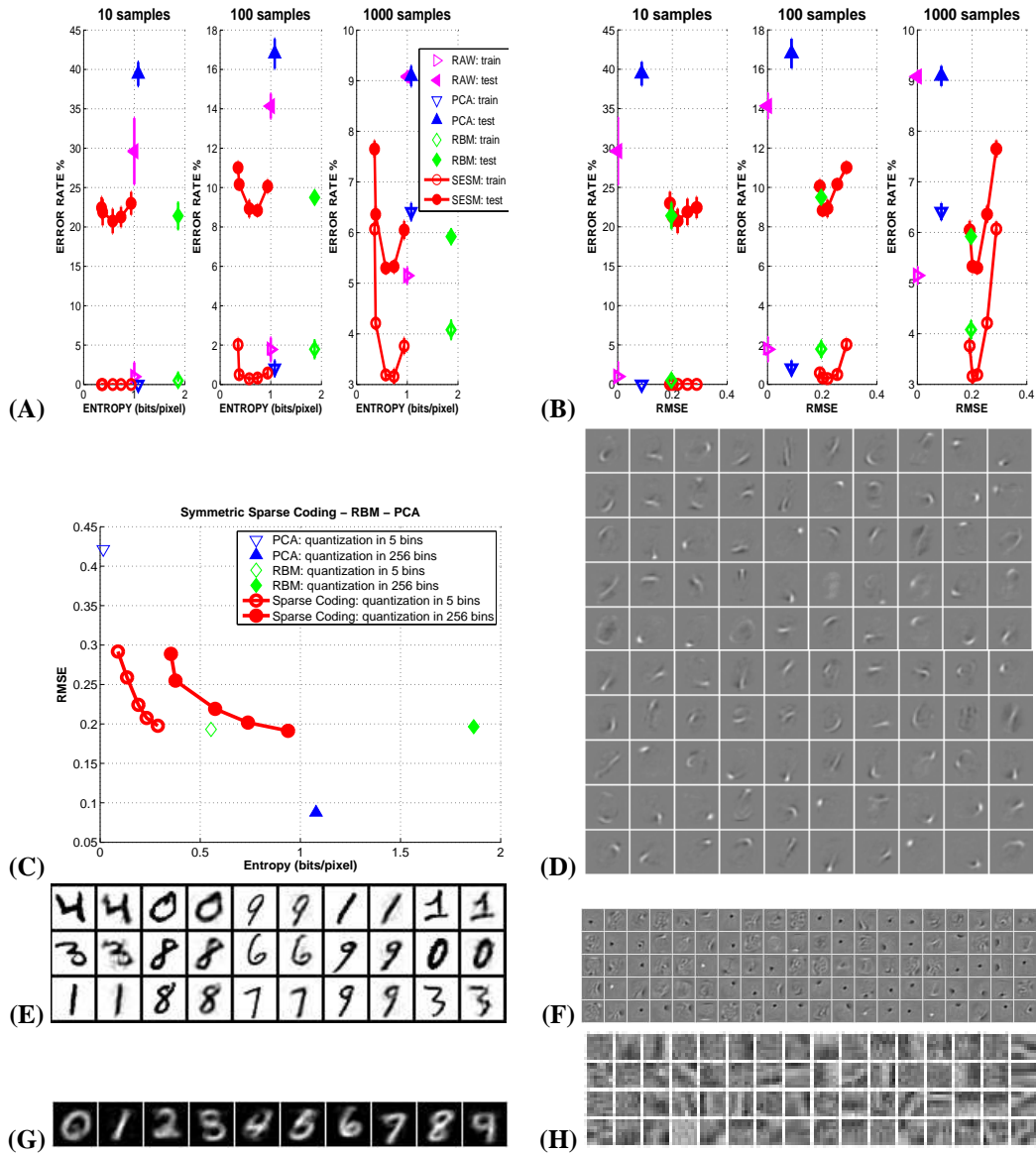

Figure 1: **(A)**-**(B)** Error rate on MNIST training (with 10, 100 and 1000 samples per class) and test set produced by a linear classifier trained on the codes produced by SESM, RBM, and PCA. The entropy and RMSE refers to a quantization into 256 bins. The comparison has been extended also to the same classifier trained on raw pixel data (showing the advantage of extracting features). The error bars refer to 1 std. dev. of the error rate for 10 random choices of training datasets (same splits for all methods). The parameter $\alpha_s$ in eq. 8 takes values: 1, 0.5, 0.2, 0.1, 0.05. **(C)** Comparison between SESM, RBM, and PCA when quantizing the code into 5 and 256 bins. **(D)** Random selection from the 200 linear filters that were learned by SESM ($\alpha_s = 0.2$). **(E)** Some pairs of original and reconstructed digit from the code produced by the encoder in SESM (feed-forward propagation through encoder and decoder). **(F)** Random selection of filters learned by RBM. **(G)** Back-projection in image space of the filters learned in the second stage of the hierarchical feature extractor. The second stage was trained on the non linearly transformed codes produced by the first stage machine. The back-projection has been performed by using a 1-of-10 code in the second stage machine, and propagating this through the second stage decoder and first stage decoder. The filters at the second stage discover the class-prototypes (manually ordered for visual convenience) even though no class label was ever used during training. **(H)** Feature extraction from 8x8 natural image patches: some filters that were learned.

when the number of training samples is small), 2) RBM performance is competitive overall when few training samples are available, 3) the best performance is achieved by SESM for a sparsity level which trades off RMSE for entropy (overall for large training sets), 4) the method with the best RMSE is *not* the one with lowest error rate, 5) compared to a SESM having the same error rate RBM is more costly in terms of entropy.

# 5 Experiments

This section describes some experiments we have done with SESM. The coefficient $\alpha_e$ in equation 8 has always been set equal to 1, and the gain in the logistic have been set equal to 7 in order to achieve a quasi-binary coding. The parameter $\alpha_s$ has to be set by cross-validation to a value which depends on the level of sparsity required by the specific application.

## 5.1 Handwritten Digits

Fig. 1-B/E shows the result of training a SESM with $\alpha_s$ is equal to 0.2. Training was performed on 20000 digits scaled between 0 and 1, by setting $\alpha_r$ to 0.0004 (in equation 8) with a learning rate equal to 0.025 (decreased exponentially). Filters detect the strokes that can be combined to form a digit. Even if the code unit activation has a very sparse distribution, reconstructions are very good (no minimization in code space was performed).

### 5.1.1 Hierarchical Features

A hierarchical feature extractor can be trained layer-by-layer similarly to what has been proposed in [19, 1] for training deep belief nets (DBNs). We have trained a second (higher) stage machine on the non linearly transformed codes produced by the first (lower) stage machine described in the previous example. We used just 20000 codes to produce a higher level representation with just 10 components. Since we aimed to find a 1-of-10 code we increased the sparsity level (in the second stage machine) by setting $\alpha_s$ to 1. Despite the completely *unsupervised* training procedure, the feature detectors in the second stage machine look like digit prototypes as can be seen in fig. 1-G. The hierarchical unsupervised feature extractor is able to capture higher order correlations among the input pixel intensities, and to discover the highly non-linear mapping from raw pixel data to the class labels. Changing the random initialization can sometimes lead to the discover of two different shapes of "9" without a unit encoding the "4", for instance. Nevertheless, results are qualitatively very similar to this one. For comparison, when training a DBN, prototypes are not recovered because the learned code is distributed among units.

## 5.2 Natural Image Patches

A SESM with about the same set up was trained on a dataset of 30000 8x8 natural image patches randomly extracted from the Berkeley segmentation dataset [20]. The input images were simply scaled down to the range $[0, 1.7]$, without even subtracting the mean. We have considered a 2 times overcomplete code with 128 units. The parameters $\alpha_s$, $\alpha_r$ and the learning rate were set to 0.4, 0.025, and 0.001 respectively. Some filters are localized Gabor-like edge detectors in different positions and orientations, other are more global, and some encode the mean value (see fig. 1-H).

# 6 Conclusions

There are two strategies to train unsupervised machines: 1) having a contrastive term in the loss function minimized during training, 2) constraining the internal representation in such a way that training samples can be better reconstructed than other points in input space. We have shown that RBM, which falls in the first class of methods, is particularly robust to channel noise, it achieves very low RMSE and good recognition rate. We have also proposed a novel symmetric sparse encoding method following the second strategy which: is particularly efficient to train, has fast inference, works without requiring any withening or even mean removal from the input, can provide the best recognition performance and trade-off between entropy/RMSE, and can be easily extended to a hierarchy discovering hidden structure in the data. We have proposed an evaluation protocol to compare different machines which is based on RMSE, entropy and, eventually, error rate when also

labels are available. Interestingly, the machine achieving the best performance in classification is the one with the best trade-off between reconstruction error and entropy. A future avenue of work is to understand the reasons for this "coincidence", and deeper connections between these two strategies.

**Acknowledgments**
We wish to thank Jonathan Goodman, Geoffrey Hinton, and Yoshua Bengio for helpful discussions. This work was supported in part by NSF grant IIS-0535166 "toward category-level object recognition", NSF ITR-0325463 "new directions in predictive learning", and ONR grant N00014-07-1-0535 "integration and representation of high dimensional data".

# References

[1] G.E. Hinton and R. R Salakhutdinov. Reducing the dimensionality of data with neural networks. *Science*, 313(5786):504–507, 2006.

[2] Y. Bengio, P. Lamblin, D. Popovici, and H. Larochelle. Greedy layer-wise training of deep networks. In *NIPS*, 2006.

[3] M. Ranzato, C. Poultney, S. Chopra, and Y. LeCun. Efficient learning of sparse representations with an energy-based model. In *NIPS 2006*. MIT Press, 2006.

[4] Y. Bengio and Y. LeCun. Scaling learning algorithms towars ai. In D. DeCoste L. Bottou, O. Chapelle and J. Weston, editors, *Large-Scale Kernel Machines*. MIT Press, 2007.

[5] M. Ranzato, Y. Boureau, S. Chopra, and Y. LeCun. A unified energy-based framework for unsupervised learning. In *Proc. Conference on AI and Statistics (AI-Stats)*, 2007.

[6] E. Doi, D. C. Balcan, and M. S. Lewicki. A theoretical analysis of robust coding over noisy overcomplete channels. In *NIPS*. MIT Press, 2006.

[7] Y. W. Teh, M. Welling, S. Osindero, and G. E. Hinton. Energy-based models for sparse overcomplete representations. *Journal of Machine Learning Research*, 4:1235–1260, 2003.

[8] B. A. Olshausen and D. J. Field. Sparse coding with an overcomplete basis set: a strategy employed by v1? *Vision Research*, 37:3311–3325, 1997.

[9] D. D. Lee and H. S. Seung. Learning the parts of objects by non-negative matrix factorization. *Nature*, 401:788–791, 1999.

[10] G.E. Hinton. Training products of experts by minimizing contrastive divergence. *Neural Computation*, 14:1771–1800, 2002.

[11] P. Lennie. The cost of cortical computation. *Current biology*, 13:493–497, 2003.

[12] J.F. Murray and K. Kreutz-Delgado. Learning sparse overcomplete codes for images. *The Journal of VLSI Signal Processing*, 45:97–110, 2008.

[13] G.E. Hinton and R.S. Zemel. Autoencoders, minimum description length, and helmholtz free energy. In *NIPS*, 1994.

[14] G.E. Hinton, P. Dayan, and M. Revow. Modeling the manifolds of images of handwritten digits. *IEEE Transactions on Neural Networks*, 8:65–74, 1997.

[15] M.S. Lewicki and T.J. Sejnowski. Learning overcomplete representations. *Neural Computation*, 12:337–365, 2000.

[16] Y. LeCun, S. Chopra, R. Hadsell, M. Ranzato, and F.J. Huang. A tutorial on energy-based learning. In G. Bakir and al.., editors, *Predicting Structured Data*. MIT Press, 2006.

[17] P. Sallee and B.A. Olshausen. Learning sparse multiscale image representations. In *NIPS*. MIT Press, 2002.

[18] http://yann.lecun.com/exdb/mnist/.

[19] G.E. Hinton, S. Osindero, and Y.-W. Teh. A fast learning algorithm for deep belief nets. *Neural Computation*, 18:1527–1554, 2006.

[20] http://www.cs.berkeley.edu/projects/vision/grouping/segbench/.

